# Compositionality, MDL Priors, and Object Recognition

Elie Bienenstock (elie@dam.brown.edu)
Stuart Geman (geman@dam.brown.edu)
Daniel Potter (dfp@dam.brown.edu)
Division of Applied Mathematics,
Brown University, Providence, RI 02912 USA

## Abstract

Images are ambiguous at each of many levels of a contextual hierarchy. Nevertheless, the high-level interpretation of most scenes is unambiguous, as evidenced by the superior performance of humans. This observation argues for global vision models, such as deformable templates. Unfortunately, such models are computationally intractable for unconstrained problems. We propose a compositional model in which primitives are recursively composed, subject to syntactic restrictions, to form tree-structured objects and object groupings. Ambiguity is propagated up the hierarchy in the form of multiple interpretations, which are later resolved by a Bayesian, equivalently minimum-description-length, cost functional.

## 1  Bayesian decision theory and compositionality

In his Essay on Probability, Laplace (1812) devotes a short chapter—his "Sixth Principle"—to what we call today the Bayesian decision rule. Laplace observes that we interpret a "regular combination," e.g., an arrangement of objects that displays some particular symmetry, as having resulted from a "regular cause" rather than arisen by chance. It is not, he argues, that a symmetric configuration is less likely to happen by chance than another arrangement. Rather, it is that among all possible combinations, which are equally favored by chance, there are very few of the regular type: *"On a table we see letters arranged in this order,* Constantinople, *and we judge that this arrangement is not the result of chance, not because it is less possible than the others, for if this word were not employed in any language*

*we should not suspect it came from any particular cause, but this word being in use amongst us, it is incomparably more probable that some person has thus arranged the aforesaid letters than that this arrangement is due to chance."* In this example, regularity is not a mathematical symmetry. Rather, it is a convention shared among language users, whereby *Constantinople* is a word, whereas *Ipctneolnosant,* a string containing the same letters but arranged in a random order, is not.

Central in Laplace's argument is the observation that the number of words in the language is smaller, indeed "incomparably" smaller, than the number of possible arrangements of letters. Indeed, if the collection of 14-letter words in a language made up, say, half of all 14-letter strings—a rich language indeed—we would, upon seeing the string *Constantinople* on the table, be far less inclined to deem it a word, and far more inclined to accept it as a possible coincidence. The sparseness of allowed combinations can be observed at all linguistic articulations (phonetic-syllabic, syllabic-lexical, lexical-syntactic, syntactic-pragmatic, to use broadly defined levels), and may be viewed as a form of *redundancy*—by analogy to error-correcting codes. This redundancy was likely devised by evolution to ensure efficient communication in spite of the ambiguity of elementary speech signals. The hierarchical compositional structure of natural visual scenes can also be thought of as redundant: the rules that govern the composition of edge elements into object boundaries, of intensities into surfaces etc., all the way to the assembly of 2-D projections of named objects, amount to a collection of drastic combinatorial restrictions. Arguably, this is why in all but a few—generally hand-crafted—cases, natural images have a unique high-level interpretation in spite of pervasive low-level ambiguity—this being amply demonstrated by the performances of our brains.

In sum, compositionality appears to be a fundamental aspect of cognition (see also von der Malsburg 1981, 1987; Fodor and Pylyshyn 1988; Bienenstock, 1991, 1994, 1996; Bienenstock and Geman 1995). We propose here to account for mental computation in general and scene interpretation in particular in terms of *elementary composition operations,* and describe a mathematical framework that we have developed to this effect. The present description is a cursory one, and some notions are illustrated on two simple examples rather than formally defined—for a detailed account, see Geman et al. (1996), Potter (1997). The *binary-image* example refers to an $N \times N$ array of binary-valued pixels, while the *Laplace-Table* example refers to a one-dimensional array of length $N$, where each position can be filled with one of the 26 letters of the alphabet or remain blank.

## 2  Labels and composition rules

The *objects* operated upon are denoted $\omega_i, i = 1, 2, \ldots, k$. Each *composite* object $\omega$ carries a *label, l = L(\omega)*, and the list of its constituents, $(\omega_1, \omega_2, \cdots)$. These uniquely determine $\omega$, so we write $\omega = l(\omega_1, \omega_2, \cdots)$. A *scene S* is a collection of *primitive* objects. In the binary-image case, a scene $S$ consists of a collection of black pixels in the $N \times N$ array. All these primitives carry the same label, $L(\omega) = p$ (for "Point"), and a parameter $\pi(\omega)$ which is the position in the image. In Laplace's Table, a scene $S$ consists of an arrangement of characters on the table. There are 26 primitive labels, "A","B",...,"Z", and the parameter of a primitive $\omega$ is its position $1 \leq \pi(\omega) \leq N$ (all primitives in such a scene must have different positions).

An example of a composite $\omega$ in the binary-image case is an arrangement composed

of a black pixel at any position except on the rightmost column and another black pixel to the immediate right of the first one. The label is "Horizontal Linelet," denoted $L(\omega) = hl$, and there are $N(N - 1)$ possible horizontal linelets. Another non-primitive label, "Vertical Linelet," or $vl$, is defined analogously. An example of a composite $\omega$ for Laplace's Table is an arrangement of 14 neighboring primitives carrying the labels "$C$", "$O$", "$N$", "$S$",$\dots$, "$E$" in that order, wherever that arrangement will fit. We then have $L(\omega) = Constantinople$, and there are $N - 13$ possible Constantinople objects.

The *composition rule* for label type $l$ consists of a *binding function*, $B_l$, and a set of allowed binding-function values, or *binding support*, $S_l$: denoting by $\Omega$ the set of *all* objects in the model, we have, for any $\omega_1, \dots, \omega_k \in \Omega$, $B_l(\omega_1, \dots, \omega_k) \in S_l \Leftrightarrow l(\omega_1, \dots, \omega_k) \in \Omega$. In the binary-image example, $B_{hl}(\omega_1, \omega_2) = B_{vl}(\omega_1, \omega_2) = (L(\omega_1), L(\omega_2), \pi(\omega_2) - \pi(\omega_1))$, $S_{hl} = \{(p, p, (1, 0))\}$ and $S_{vl} = \{(p, p, (0, 1))\}$ define the $hl$- and $vl$-composition rules, $p + p \to hl$ and $p + p \to vl$. In Laplace's Table, $C + O + \dots + E \to Constantinople$ is an example of a 14-ary composition rule, where we must check the label and position of each constituent. One way to define the binding function and support for this rule is: $B(\omega_1, \dots, \omega_{14}) = (L(\omega_1), \dots, L(\omega_{14}), \pi(\omega_2) - \pi(\omega_1), \pi(\omega_3) - \pi(\omega_1), \dots, \pi(\omega_{14}) - \pi(\omega_1))$ and $S = (C, \dots, E, 1, 2, \dots, 13)$.

We now introduce *recursive* labels and composition rules: the label of the composite object is identical to the label of one or more of its constituents, and the rule may be applied an arbitrary number of times, to yield objects of arbitrary complexity. In the binary-image case, we use a recursive label $c$, for *Curve*, and an associated binding function which creates objects of the form $hl + p \to c$, $vl + p \to c$, $c + p \to c$, $p + hl \to c$, $p + vl \to c$, $p + c \to c$, and $c + c \to c$. The reader may easily fill in the details, i.e., define a binding function and binding support which result in "$c$"-objects being precisely curves in the image, where a curve is of length at least 3 and may be self-intersecting. In the previous examples, primitives were composed into compositions; here compositions are further composed into more complex compositions. In general, an object $\omega$ is a *labeled tree*, where each vertex carries the name of an object, and each leaf is associated with a primitive (the association is not necessarily one-to-one, as in the case of a self-intersecting curve).

Let $\mathcal{M}$ be a *model*—i.e., a collection of labels with their binding functions and binding supports—and $\Omega$ the set of all objects in $\mathcal{M}$. We say that object $\omega \in \Omega$ *covers* $S$ if $S$ is precisely the set of primitives that make up $\omega$'s leaves. An *interpretation* $I$ of $S$ is any finite collection of objects in $\Omega$ such that the union of the sets of primitives they cover is $S$. We use the convention that, for all $\mathcal{M}$ and $S$, $I_0$ denotes the *trivial* interpretation, defined as the collection of (unbound) primitives in $S$. In most cases of interest, a model $\mathcal{M}$ will allow many interpretations for a scene $S$. For instance, given a long curve in the binary-image model, there will be many ways to recursively construct a "$c$"-labeled tree that covers exactly that curve.

## 3   The MDL formulation

In Laplace's Table, a scene consisting of the string *Constantinople* admits, in addition to $I_0$, the interpretation $I_1 = \{\omega_1\}$, where $\omega_1$ is a "*Constantinople*"-object. We wish to define a probability distribution $D$ on interpretations such that $D(I_1) >> D(I_0)$, in order to realize Laplace's "incomparably more probable". Our

definition of $D$ will be motivated by the following use of the Minimum Description Length (MDL) principle (Rissanen 1989). Consider a scene $S$ and pretend we want to transmit $S$ as quickly as possible through a noiseless channel, hence we seek to *encode* it as efficiently as possible, i.e., with the shortest possible binary code $c$. We can always use the trivial interpretation $I_0$: the codeword $c(I_0)$ is a mere list of $n$ locations in $S$. We need not specify labels, since there is only one primitive label in this example. The length, or *cost*, of this code for $S$ is $|c(I_0)| = n \log_2(N^2)$.

Now however we want to take advantage of regularities, in the sense of Laplace, that we *expect* to be present in $S$. We are specifically interested in compositional regularities, where some arrangements that occur more frequently than by chance can be interpreted *advantageously* using an appropriate compositional model $\mathcal{M}$. Interpretation $I$ is advantageous if $|c(I)| < |c(I_0)|$. An example in the binary-image case is a linelet scene $S$. The trivial encoding of this scene costs us $|c(I_0)| = 2[\log_2 3 + \log_2(N^2)]$ bits, whereas the cost of the compositional interpretation $I_1 = \{\omega_1\}$ is $|c(I_1)| = \log_2 3 + \log_2(N(N-1))$, where $\omega_1$ is an $hl$ or $vl$ object, as the case may be. The first $\log_2 3$ bits encode the label $L(\omega_1) \in \{p, hl, vl\}$, and the rest encodes the position in the image. The compositional $\{p, hl, vl\}$ model is therefore advantageous for a linelet scene, since $I_1$ affords us a gain in encoding cost of about $2 \log_2 N$ bits.

In general, the gain realized by encoding $\{\omega\} = \{l(\omega_1, \omega_2)\}$ instead of $\{\omega_1, \omega_2\}$ may be viewed as a *binding energy*, measuring the affinity that $\omega_1$ and $\omega_2$ exhibit for each other as they assemble into $\omega$. This binding energy is $\mathcal{E}_l = |c(\omega_1)| + |c(\omega_2)| - |c(l(\omega_1, \omega_2))|$, and an efficient $\mathcal{M}$ is one that contains *judiciously chosen* cost-saving composition rules. In effect, if, say, linelets were very rare, we would be better off with the trivial model. The inclusion of non-primitive labels would force us to add at least one bit to the code of every object—to specify its label—and this would increase the *average* encoding cost, since the infrequent use of non-primitive labels would not balance the extra small cost incurred on primitives. In practical applications, the construction of a sound $\mathcal{M}$ is no trivial issue. Note however the simple rationale for including a rule such as $p + p \to hl$. Giving ourselves the label $hl$ renders redundant the independent encoding of the positions of horizontally adjacent pixels. In general, a good model should allow one to hierarchically compose with each other *frequently occurring* arrangements of objects.

This use of MDL leads in a straightforward way to an equivalent Bayesian formulation. Setting $P'(\omega) = 2^{-|c(\omega)|} / \sum_{\omega' \in \Omega} 2^{-|c(\omega')|}$ yields a probability distribution $P'$ on $\Omega$ for which $c$ is approximately a Shannon code (Cover and Thomas 1991). With this definition, the decision to include the label $hl$—or the label *Constantinople*—would be viewed, in principle, as a statement about the prior probability of finding horizontal linelets—or *Constantinople* strings—in the scene to be interpreted.

## 4   The observable-measure formulation

The MDL formulation however has a number of shortcomings; in particular, computing the binding energy for composite objects can be problematic. We outline now an alternative approach (Geman et al. 1996, Potter 1997), where a probability distribution $P(\omega)$ on $\Omega$ is defined through a family of *observable measures* $Q_l$. These measures assign probabilities to each possible binding-function value, $s \in S_l$, and also to the primitives. We require $\sum_{l \in \mathcal{M}} \sum_{s \in S_l} Q_l(s) = 1$, where the notion of binding function has been extended to primitives via $B_{prim}(\omega) = \pi(\omega)$ for primitive

$\omega$. The probabilities induced on $\Omega$ by $Q_l$ are given by $P(\omega) = Q_{prim}(B_{prim}(\omega))$ for a primitive $\omega$, and $P(\omega) = Q_l(B_l(\omega_1, \omega_2))P^2(\omega_1, \omega_2|B_l(\omega_1, \omega_2))$ for a composite object $\omega = l(\omega_1, \omega_2)$.[1] Here $P^2 = P \times P$ is the product probability, i.e., the *free*, or *not-bound*, distribution for the pair $(\omega_1, \omega_2) \in \Omega^2$. For instance, with $C + \cdots + E \rightarrow$ *Constantinople*, $P^{14}(\omega_1, \omega_2, \ldots, \omega_{14}|B_{Cons...}(\omega_1, \ldots, \omega_{14}) = (C, O, \cdots, 13)) $ is the conditional probability of observing a *particular* string *Constantinople*, under the free distribution, given that $(\omega_1, \ldots, \omega_{14})$ constitutes such a string. With the reasonable assumption that, under $Q$, primitives are uniformly distributed over the table, this conditional probability is simply the inverse of the number of possible *Constantinople* strings, i.e., $1/(N-13)$.

The binding energy, defined, by analogy to the MDL approach, as $\mathcal{E}_l = \log_2(P(\omega)/(P(\omega_1)P(\omega_2)))$, now becomes $\mathcal{E}_l = \log_2(Q_l(B_l(\omega_1, \omega_2))) - \log_2(P \times P(B_l(\omega_1, \omega_2)))$. Finally, if $\mathcal{I}$ is the collection of all finite interpretations $I \subset \Omega$, we define the probability of $I \in \mathcal{I}$ as $D(I) = \Pi_{\omega \in I}P(\omega)/Z$, with $Z = \sum_{I' \in \mathcal{I}} \Pi_{\omega \in I'} P(\omega)$. Thus, the probability of an interpretation containing several free objects is obtained by assuming that these objects occurred in the scene independently of each other. Given a scene $S$, recognition is formulated as the task of maximizing $D$ over all the $I$'s in $\mathcal{I}$ that are interpretations of $S$.

We now illustrate the use of $D$ on our two examples. In the binary-image example with model $\mathcal{M} = \{p, hl, vl\}$, we use a parameter $q, 0 \le q \le 1$, to adjust the prior probability of linelets. Thus, $Q_{prim}(B_{prim}(\omega)) = (1-q)/N^2$ for primitives, and $Q_{hl}((p, p, 0, 1)) = Q_{vl}((p, p, 1, 0)) = q/2$ for linelets. It is easily seen that regardless of the normalizing constant $Z$, the binding energy of two adjacent pixels into a linelet is $\mathcal{E}_{hl} = \mathcal{E}_{vl} = \log_2(q/2) - \log_2[\frac{(1-q)^2}{N^4}N(N-1)]$. Interestingly, as long as $q \ne 0$ and $q \ne 1$, the binding energy, for large $N$, is approximately $2\log_2 N$, which is independent of $q$. Thus, the linelet interpretation is "incomparably" more likely than the independent occurrence of two primitives at neighboring positions. We leave it to the reader to construct a prior $P$ for the model $\{p, hl, vl, c\}$, e.g. by distributing the $Q$-mass evenly between all composition rules. Finally, in Laplace's Table, if there are $M$ equally likely non-primitive labels—say city names—and $q$ is their total mass, the binding energy for *Constantinople* is $\mathcal{E}_{Cons...} = \log_2 \frac{q}{M(N-13)} - \log_2[\frac{1-q}{26N}]^{14}$, and the "regular" cause is again "incomparably" more likely.

There are several advantages to this reformulation from codewords into probabilities using the $Q$-parameters. First, the $Q$-parameters can in principle be adjusted to better account for a particular world of images. Second, we get an explicit formula for the binding energy, (namely $\log_2(Q/P \times P)$). Of course, we need to evaluate the product probability $P \times P$, and this can be highly non-trivial—one approach is through sampling, as demonstrated in Potter (1997). Finally, this formulation is well-suited for parameter estimation: the $Q$'s, which are the parameters of the distribution $P$, are indeed observables, i.e., directly available empirically.

## 5   Concluding remarks

The approach described here was applied by X. Xing to the recognition of "on-line" handwritten characters, using a binary-image-type model as above, enriched

with higher-level labels including curved lines, straight lines, angles, crossings, T-junctions, L-junctions (right angles), and the 26 letters of the alphabet. In such a model, the search for an optimal solution cannot be done exhaustively. We experimented with a number of strategies, including a two-step algorithm which first generates *all* possible objects in the scene, and then selects the "best" objects, i.e., the objects with highest *total* binding energy, using a greedy method, to yield a final scene interpretation. (The total binding energy of $\omega$ is the sum of the binding energies $\mathcal{E}_l$ over all the composition rules $l$ used in the composition of $\omega$. Equivalently, the total binding energy is the log-likelihood ratio $\log_2(P(\omega)/\Pi_i P(\omega_i))$, where the product is taken over all the primitives $\omega_i$ covered by $\omega$.)

The first step of the algorithm typically results in high-level objects partly overlapping on the set of primitives they cover, i.e., competing for the interpretation of shared primitives. Ambiguity is thus propagated in a "bottom-up" fashion. The ambiguity is resolved in the second "top-down" pass, when high-level composition rules are used to select the best compositions, at all levels including the lower ones. A detailed account of our experiments will be given elsewhere. We found the results quite encouraging, particularly in view of the potential scope of the approach. In effect, we believe that this approach is in principle capable of addressing unrestricted vision problems, where images are typically very ambiguous at lower levels for a variety of reasons—including occlusion and mutual overlap of objects—hence purely bottom-up segmentation is impractical.

Turning now to biological implications, note that dynamic binding in the nervous system has been a subject of intensive research and debate in the last decade. Most interesting in the present context is the suggestion, first clearly articulated by von der Malsburg (1981), that composition may be performed thanks to a dual mechanism of accurate synchronization of spiking activity—not necessarily relying on periodic firing—and fast reversible synaptic plasticity. If there is some neurobiological truth to the model described in the present paper, the binding mechanism proposed by von der Malsburg would appear to be an attractive implementation. In effect, the use of fine temporal structure of neural activity opens up a large realm of possible high-order codes in networks of neurons.

In the present model, constituents always bind *in the service* of a new object, an operation one may refer to as *triangular binding*. Composite objects can engage in further composition, thus giving rise to arbitrarily deep tree-structured constructs. Physiological evidence of triangular binding in the visual system can be found in Sillito et al. (1994); Damasio (1989) describes an approach derived from neuroanatomical data and lesion studies that is largely consistent with the formalism described here.

An important requirement for the neural representation of the tree-structured objects used in our model is that the doing and undoing of links operating on some constituents, say $\omega_1$ and $\omega_2$, while affecting in some *useful* way the high-order patterns that represent these objects, leaves these patterns, as representations of $\omega_1$ and $\omega_2$, intact. A family of biologically plausible patterns that would appear to satisfy this requirement is provided by *synfire* patterns (Abeles 1991). We hypothesized elsewhere (Bienenstock 1991, 1994, 1996) that synfire chains could be dynamically bound via weak synaptic couplings; such couplings would synchronize the wave-like activities of two synfire chains, in much the same way as coupled oscillators lock

their phases. Recursiveness of compositionality could, in principle, arise from the further binding of these composite structures.

## Acknowledgements

Supported by the Army Research Office (DAAL03-92-G-0115), the National Science Foundation (DMS-9217655), and the Office of Naval Research (N00014-96-1-0647).

## Footnotes

[1]This is actually an implicit definition. Under reasonable conditions, it is well defined— see Geman et al. (1996).

## References

Abeles, M. (1991) *Corticonics: Neuronal circuits of the cerebral cortex*, Cambridge University Press.

Bienenstock, E. (1991) Notes on the growth of a composition machine, in *Proceedings of the Royaumont Interdisciplinary Workshop on Compositionality in Cognition and Neural Networks—I*, D. Andler, E. Bienenstock, and B. Laks, Eds., pp. 25–43. (1994) A Model of Neocortex. *Network: Computation in Neural Systems*, 6:179–224. (1996) Composition, In *Brain Theory: Biological Basis and Computational Principles*, A. Aertsen and V. Braitenberg eds., Elsevier, pp 269–300.

Bienenstock, E., and Geman, S. (1995) Compositionality in Neural Systems, In *The Handbook of Brain Theory and Neural Networks*, M.A. Arbib ed., M.I.T./Bradford Press, pp 223–226.

Cover, T.M., and Thomas, J.A. (1991) *Elements of Information Theory*, Wiley and Sons, New York.

Damasio, A. R. (1989) Time-locked multiregional retroactivation: a systems-level proposal for the neural substrates of recall and recognition, *Cognition*, 33:25–62.

Fodor, J.A., and Pylyshyn, Z.W. (1988) Connectionism and cognitive architecture: a critical analysis, *Cognition*, 28:3–71.

Geman, S., Potter, D., and Chi, Z. (1996) *Compositional Systems*, Technical Report, Division of Applied Mathematics, Brown University.

Laplace, P.S. (1812) *Esssai philosophique sur les probabilités*. Translation of Truscott and Emory, New York, 1902.

Potter, D. (1997) *Compositional Pattern Recognition*, PhD Thesis, Division of Applied Mathematics, Brown University, In preparation.

Rissanen, J. (1989) *Stochastic Complexity in Statistical Inquiry* World Scientific Co, Singapore.

Sillito, A.M., Jones, H.E, Gerstein, G.L., and West, D.C. (1994) Feature-linked synchronization of thalamic relay cell firing induced by feedback from the visual cortex, *Nature*, 369: 479-482

von der Malsburg, C. (1981) *The correlation theory of brain function*. Internal report 81-2, Max-Planck Institute for Biophysical Chemistry, Dept. of Neurobiology, Göttingen, Germany. (1987) Synaptic plasticity as a basis of brain organization, in *The Neural and Molecular Bases of Learning* (J.P. Changeux and M. Konishi, Eds.), John Wiley and Sons, pp. 411–432.